# Neural Network Implementation of Admission Control

**Rodolfo A. Milito, Isabelle Guyon, and Sara A. Solla**
AT&T Bell Laboratories, Crawfords Corner Rd., Holmdel, NJ 07733

## Abstract

A feedforward layered network implements a mapping required to control an unknown stochastic nonlinear dynamical system. Training is based on a novel approach that combines stochastic approximation ideas with back-propagation. The method is applied to control admission into a queueing system operating in a time-varying environment.

## 1 INTRODUCTION

A controller for a discrete-time dynamical system must provide, at time $t_n$, a value $u_n$ for the control variable. Information about the state of the system when such decision is made is available through the observable $y_n$. The value $u_n$ is determined on the basis of the current observation $y_n$ and the preceding control action $u_{n-1}$. Given the information $I_n = (y_n, u_{n-1})$, the controller implements a mapping $I_n \rightarrow u_n$.

Open-loop controllers suffice in static situations which require a single-valued control policy $u^*$: a constant mapping $I_n \rightarrow u^*$, regardless of $I_n$. Closed-loop controllers provide a dynamic control action $u_n$, determined by the available information $I_n$. This work addresses the question of training a neural network to implement a general mapping $I_n \rightarrow u_n$.

The problem that arises is the lack of training patterns: the appropriate value $u_n$ for a given input $I_n$ is not known. The quality of a given control policy can only be assessed by using it to control the system, and monitoring system performance. The sensitivity of the performance to variations in the control policy cannot be investigated analytically, since the system is unknown. We show that such sensitivity can be estimated within the standard framework of stochastic approximation. The usual back-propagation algorithm is used to determine the sensitivity of the output $u_n$ to variations in the parameters $\mathbf{W}$ of the network, which can thus be adjusted so as to improve system performance.

The advantage of a neural network as a closed-loop controller resides in its ability to accept inputs $(I_n, I_{n-1}, ..., I_{n-p})$. The additional $p$ time steps into the past provide information about the history of the controlled system. As demonstrated here, neural network controllers can capture regularities in the structure of time-varying environments, and are particularly powerful for tracking time variations driven by stationary stochastic processes.

## 2   CONTROL OF STOCHASTIC DYNAMICAL SYSTEMS

Consider a dynamical system for which the state $x_n$ is updated at discrete times $t_n = n\,\delta$. The control input $u_n$ in effect at time $t_n$ affects the dynamical evolution, and

$$x_{n+1} = f(x_n, u_n, \xi_n). \tag{2.1}$$

Here $\{\xi_n\}$ is a stochastic process which models the intrinsic randomness of the system as well as external, unmeasurable disturbances. The variable $x_n$ is not accessible to direct measurement, and knowledge about the state of the system is limited to the observable

$$y_n = h(x_n). \tag{2.2}$$

Our goal is to design a neural network controller which produces a specific value $u_n$ for the control variable to be applied at time $t_n$, given the available information $I_n \equiv (y_n, u_{n-1})$.

In order to design a controller which implements the appropriate *control policy* $I_n \rightarrow u_n$, a specification of the purpose of controlling the dynamical system is needed. There is typically a function of the observable,

$$J_n = H(y_n), \tag{2.3}$$

which measures system performance. It follows from Eqs. (2.1)-(2.3) that the composition $G = H \circ h \circ f$ determines

$$J_n = G(x_{n-1}, u_{n-1}, \xi_{n-1}), \tag{2.4}$$

a function of the state $x$ of the system, the control variable $u$, and the stochastic variable $\xi$. The quantity of interest is the expectation value of the system performance,

$$\langle J_n \rangle = \langle H(y_n) \rangle_\xi, \tag{2.5}$$

averaged with respect to $\xi$. This expectation value can be estimated by the long-run average

$$\bar{J}_N = \frac{1}{N} \sum_{n=1}^{N} H(y_n), \tag{2.6}$$

since for an ergodic system $\bar{J}_N \rightarrow \langle J_n \rangle$ as $N \rightarrow \infty$. The goal of the controller is to generate a sequence $\{u_n\}$, $1 \le n \le N$ of control values, such that the average performance $\langle J_n \rangle$ stabilizes to a desired value $J^*$.

The parameters $W$ of the neural network are thus to be adapted so as to minimize a cost function

$$E(\mathbf{W}) = \frac{1}{2} \, (<J_n> - J^*)^2. \tag{2.7}$$

The dependence of $E(\mathbf{W})$ on $\mathbf{W}$ is implicit: the value of $<J_n>$ depends on the controlling sequence $\{u_n\}$, which depends on the parameters $\mathbf{W}$ of the neural network.

On-line training proceeds through a gradient descent update

$$\mathbf{W}_{n+1} = \mathbf{W}_n - \eta \, \nabla_{\mathbf{W}} E_n(\mathbf{W}), \tag{2.8}$$

towards the minimization of the instantaneous deviation

$$E_n(\mathbf{W}) = \frac{1}{2} \, (J_{n+1} - J^*)^2. \tag{2.9}$$

There is no specified target for the output $u_n$ that the controller is expected to provide in response to the input $I_n = (y_n, u_{n-1})$. The output $u_n$ can thus be considered as a variable $u$, which controls the subsequent performance: $J_{n+1} = G(x_n, u, \xi_n)$, as follows from Eq. (2.4). Then

$$\nabla_{\mathbf{W}} E_n(\mathbf{W}) = \frac{\partial E_n(\mathbf{W})}{\partial J_{n+1}} \, \frac{\partial G}{\partial u} \Big|_{u=u_n} \, \nabla_{\mathbf{W}} u \tag{2.10}$$

$$= (J_{n+1} - J^*) \, \frac{\partial G}{\partial u} \Big|_{u=u_n} \, \nabla_{\mathbf{W}} u.$$

The factor $\nabla_{\mathbf{W}} u$ measures the sensitivity of the output of the neural network controller to changes in the internal parameters $\mathbf{W}$: at fixed input $I_n$, the output $u_n$ is a function only of the network parameters $\mathbf{W}$. The gradient of this scalar function is easily computed using the standard back-propagation algorithm ( Rumelhart et al, 1986).

The factor $\partial G / \partial u$ measures the sensitivity of the system performance $J_{n+1}$ to changes in the control variable. The information about the system needed to evaluate this derivative is not available: unknown are the functions $f$ which describes how $x_{n+1}$ is affected by $u_n$ at fixed $x_n$, and the function $h$ which describes how this dependence propagates to the observable $y_{n+1}$. The algorithm is rendered operational through the use of stochastic approximation (Kushner, 1971): assuming that the average system performance $<J_n>$ is a monotonically increasing function of $u$, the sign of the partial derivative $\partial <J_n>/\partial u$ is positive. Stochastic approximation amounts to neglecting the unknown fluctuations of this derivative with $u$, and approximating it by a constant positive value, which is then absorbed in a redefinition of the step size $\eta > 0$.

The on-line update rule then becomes:

$$\mathbf{W}_{n+1} = \mathbf{W}_n - \eta \, (J_{n+1} - J^*) \, \nabla_{\mathbf{W}} u_n. \tag{2.11}$$

As with stochastic approximation, the on-line gradient update uses the instantaneous gradient based on the current measurement $J_{n+1}$, rather than the gradient of the expected

value $<J_n>$, whose deviations with respect to the target $J^*$ are to be minimized. The combined use of back-propagation and stochastic approximation to evaluate $\nabla_W E_n(W)$, leading to the update rule of Eq. (2.11), provides a general and powerful learning rule for neural network controllers. The only requirement is that the average performance $<J_n>$ be indeed a monotonic function of the control variable $u$.

In the following section we illustrate the application of the algorithm to an admission controller for a traffic queueing problem. The advantage of the neural network over a standard stochastic approximation approach becomes apparent when the mapping which produces $u_n$ is used to track a time-varying environment generated by a stationary stochastic process. A straightforward extension of the approach discussed above is used to train a network to implement a mapping $(I_n, I_{n-1}, ..., I_{n-p}) \rightarrow u_n$. The additional $p$ time steps into the past provide information on the history of the controlled system, and allow the network to capture regularities in the time variations of the environment.

## 3   A TWO-TRAFFIC QUEUEING PROBLEM

Consider an admission controller for a queueing system. As depicted in Fig. 1, the system includes a server, a queue, a call admission mechanism, and a controller.

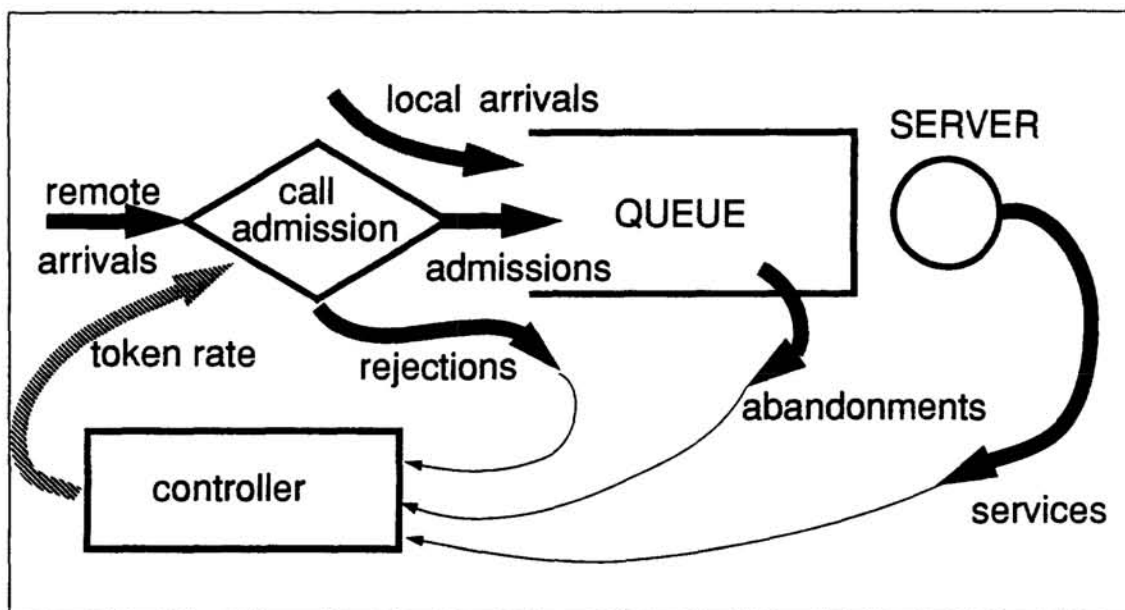

**Figure 1:** Admission controller for a two-traffic queuing problem.

The need to serve two independent traffic streams with a single server arises often in telecommunication networks. In a typical situation, in addition to remote arrivals which can be monitored at the control node, there are local arrivals whose admission to the queue can be neither monitored nor regulated. Within this limited information scenario, the controller must execute a policy that meets specified performance objectives. Such is the situation we now model.

Two streams are offered to the queueing system: remote traffic and local traffic. Both streams are Poisson, i.e., the interarrival times are independently and exponentially distributed, with mean $1/\lambda$. Calls originated by the remote stream can be controlled, by denying admission to the queue. Local calls are neither controlled nor monitored. While the arrival rate $\lambda_R$ of remote calls is fixed, the rate $\lambda_L(t)$ of local calls is time-varying. It depends on the state of a stationary Markov chain to be described later (Kleinrock, 1975).

The service time required by a call of any type is an exponentially distributed random variable, with mean $1/\mu$.

Calls that find an empty queue on arrival get immediately into service. Otherwise, they wait in queue. The service discipline is first in first out, non-idling. Every arrival is assigned a "patience threshold" $\tau$, independently drawn from a fixed but unknown distribution that characterizes customer behavior. If the waiting time in queue exceeds its "patience threshold", the call abandons.

Ideally, every incoming call should be admitted. The server, however, cannot process, on the average, more than $\mu$ calls per unit time. Whenever the offered load $\rho = [\lambda_R + \lambda_L(t)]/\mu$ approaches or exceeds 1, the queue starts to build up. Long queues result in long delays, which in turn induce heavy abandonments. To keep the abandonments within tolerable limits, it becomes necessary to reject some remote arrivals.

The call admission mechanism is implemented via a token-bank (not shown in the figure) rate control throttle (Berger, 1991). Tokens arrive at the token-bank at a deterministic rate $\lambda_T$. The token-bank is finite, and tokens that find a full bank are lost. A token is needed by a remote call to be admitted to the queue, and tokens are not reusable. Calls that find an empty token bank are rejected. Remote admissions are thus controlled through $u=\lambda_T/\lambda_R$.

Local calls are always admitted. The local arrival rate $\lambda_L(t)$ is controlled by an underlying $q$-state Markov chain, a birth-death process (Kleinrock, 1975) with transition rate $\gamma$ only between neighboring states. When the Markov chain is in state $i$, $1 \leq i \leq q$, the local arrival rate is $\lambda_L(i)$.

Complete specification of the state $x_n$ of the system at time $t_n$ would require information about number of *arrivals*, *abandonments*, and *services* for both remote and local traffic during the preceding time interval of duration $\delta = 1$, as well as *rejections* for the controllable remote traffic, and *waiting time* for every queued call. But the local traffic is not monitored, and information on arrivals and waiting times is not accessible. Thus $y_n$ only contains information about the remote traffic: the number $n_r$ of rejected calls, the number $n_a$ of abandonments, and the number $n_s$ of serviced calls since $t_{n-1}$. The information $I_n$ available at time $t_n$ also includes the preceding control action $u_{n-1}$. The controller uses $(I_n, I_{n-1}, \ldots, I_{n-p})$ to determine $u_n$.

The goal of the control policy is to admit as many calls as possible, compatible with a tolerable rate of abandonment $n_a / \lambda_R \leq \Delta$. The ratio $n_a / \lambda_R$ thus plays the role of the performance measure $J_n$, and its target value is $J^* = \Delta$. Values in excess of $\Delta$ imply an excessive number of abandonments and require stricter admission control. Values smaller than $\Delta$ are penalized if obtained at the expense of avoidable rejections.

## 4  RESULTS

All simulations reported here correspond to a server capable of handling calls at a rate of $\mu = 200$ per unit time. The remote traffic arrival rate is $\lambda_R = 100$. The local traffic arrival rate is controlled by a $q = 10$ Markov chain with $\lambda_L(i) = 20i$ for $1 \leq i \leq 10$. The offered load thus spans the range $0.6 \leq \rho \leq 1.5$, in steps of 0.1. Transition rates $\gamma = 0.1$, 1, and 10 in the Markov chain have been used to simulate slow, moderate, and rapid variations in the offered load.

The neural network controller receives inputs $(I_n, I_{n-1}, ..., I_{n-4})$ at time $t_n$ through 20 input units. A hidden layer with 6 units transmits information to the single output unit, which provides $u_n$. The bound for tolerable abandonment rate is set at $\Delta = 0.1$.

To check whether the neural network controller is capable of correct generalization, a network trained under a time-varying scenario was subject to a static one for testing. Training takes place under an offered load $\rho$ varying at a rate of $\gamma = 1$. The network is tested at $\gamma = 0$: the underlying Markov chain is frozen and $\rho$ is kept fixed for a long enough period to stabilize the control variable around a fixed value $u^*$, and obtain statistically meaningful values for $n_a$, $n_r$, and $n_s$. A careful numerical investigation of these quantities as a function of $\rho$ reveals that the neural network has developed an adequate control policy: light loads $\rho \leq 0.8$ spontaneously result in low values of $n_a$ and require no control ($u = 1.25$ guarantees ample token supply, and $n_r \approx 0$), but as $\rho$ exceeds 1, the system is controlled by decreasing the value of $u$ below 1, thus increasing $n_r$ to satisfy the requirement $n_a / \lambda_R \leq \Delta$. Detailed results of the static performance in comparison with a standard stochastic approximation approach will be reported elsewhere.

It is in the tracking of a time-varying environment that the power of the neural network controller is revealed. A network trained under a varying offered load is tested dynamically by monitoring the distribution of abandonments and rejections as the network controls an environment varying at the same rate $\gamma$ as used during training. The abandonment distribution $F_a(x) = \text{Prob} \{n_a / \lambda_R \leq x\}$, shown in Fig. 2 (a) for $\gamma = 1$, indicates that the neural network (NN) controller outperforms both stochastic approximation[1] (SA) and the uncontrolled system (UN): the probability of keeping the abandonment rate $n_r / \lambda_R$ bounded is larger for the NN controller for *all* values of the bound $x$. As for the goal of not exceeding $x = \Delta$, it is achieved with probability $F_a(\Delta) = 0.88$ by the NN, in comparison to only $F_a(\Delta) = 0.74$ with SA or $F_a(\Delta) = 0.51$ if UN. The rejection distribution $F_r(x) = \text{Prob} \{n_r / \lambda_R \leq x\}$, shown in Fig. 2 (b) for $\gamma = 1$, illustrates the stricter control policy provided by NN. Results for $\gamma = 0.1$ and $\gamma = 10$, not shown here, confirm the superiority of the control policy

developed by the neural network.

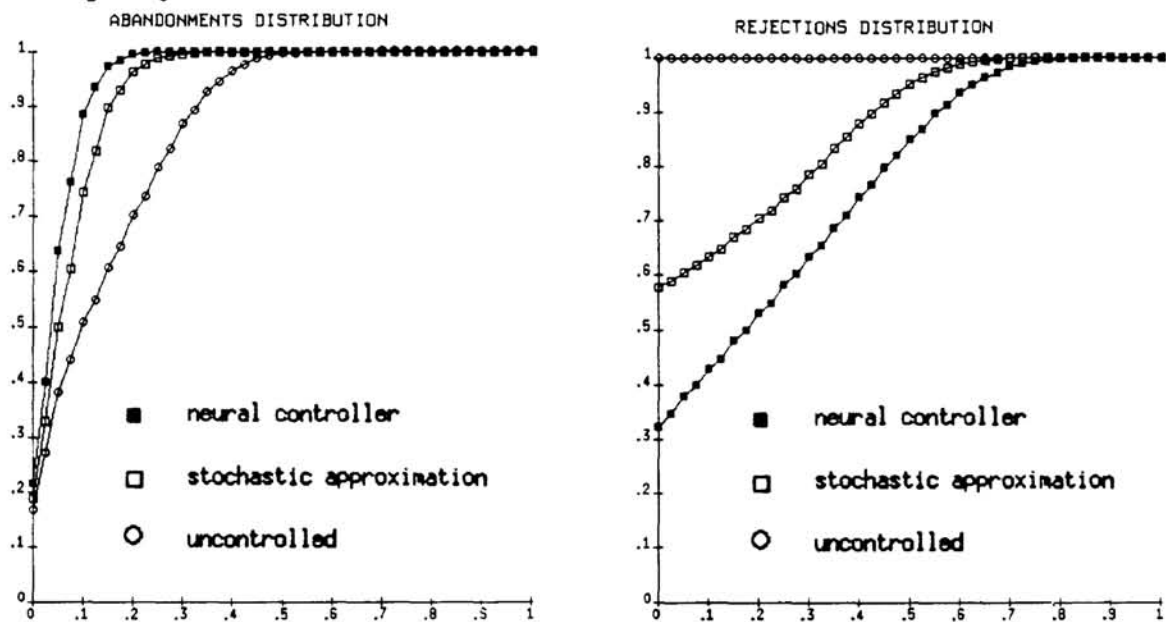

**Figure 2:** (a) Abandonment distribution $F_a(x)$, and (b) rejection distribution $F_r(x)$.

## 5  CONCLUSIONS

The control of an unknown stochastic system requires a mapping that is implemented here via a feedforward layered neural network. A novel learning rule, a blend of stochastic approximation and back-propagation, is proposed to overcome the lack of training patterns through the use of on-line performance information provided by the system under control. Satisfactorily tested for an admission control problem, the approach shows promise for a variety of applications to congestion control in telecommunication networks.

## Footnotes

[1] Stochastic approximation with a fixed gain, to enable the controller to track time-varying environments. The gain was optimized numerically.

## References

A.W. Berger, "Overload control using a rate control throttle: selecting token capacity for robustness to arrival rates", *IEEE Transactions on Automatic Control* **36**, 216-219 (1991).

H. Kushner, *Stochastic Approximation Methods for Constrained and Unconstrained Systems*, Springer Verlag (1971).

L. Kleinrock, *QUEUEING SYSTEMS Volume I: Theory*, John Wiley & Sons (1975).

D.E. Rumelhart, G.E. Hinton, and R.J. Williams, "Learning representations by back-propagating errors", *Nature* **323**, 533-536 (1986).
